# When Will a Genetic Algorithm Outperform Hill Climbing?

**Melanie Mitchell**
Santa Fe Institute
1660 Old Pecos Trail, Suite A
Santa Fe, NM 87501

**John H. Holland**
Dept. of Psychology
University of Michigan
Ann Arbor, MI 48109

**Stephanie Forrest**
Dept. of Computer Science
University of New Mexico
Albuquerque, NM 87131

## Abstract

We analyze a simple hill-climbing algorithm (RMHC) that was previously shown to outperform a genetic algorithm (GA) on a simple "Royal Road" function. We then analyze an "idealized" genetic algorithm (IGA) that is significantly faster than RMHC and that gives a lower bound for GA speed. We identify the features of the IGA that give rise to this speedup, and discuss how these features can be incorporated into a real GA.

## 1 INTRODUCTION

Our goal is to understand the class of problems for which genetic algorithms (GA) are most suited, and in particular, for which they will outperform other search algorithms. Several studies have empirically compared GAs with other search and optimization methods such as simple hill-climbing (e.g., Davis, 1991), simulated annealing (e.g., Ingber & Rosen, 1992), linear, nonlinear, and integer programming techniques, and other traditional optimization techniques (e.g., De Jong, 1975). However, such comparisons typically compare one version of the GA with a second algorithm on a single problem or set of problems, often using performance criteria which may not be appropriate. These comparisons typically do not identify the features that led to better performance by one or the other algorithm, making it hard to distill general principles from these isolated results. In this paper we look in depth at one simple hill-climbing method and an idealized form of the GA, in order to identify some general principles about when and why a GA will outperform hill climbing.

$s_1 = $ 11111111********************************************************; $c_1 = 8$
$s_2 = $ ********11111111************************************************; $c_2 = 8$
$s_3 = $ ****************11111111****************************************; $c_3 = 8$
$s_4 = $ ************************11111111********************************; $c_4 = 8$
$s_5 = $ ********************************11111111************************; $c_5 = 8$
$s_6 = $ ****************************************11111111****************; $c_6 = 8$
$s_7 = $ ************************************************11111111********; $c_7 = 8$
$s_8 = $ ********************************************************11111111; $c_8 = 8$
$s_{opt}=$11111111111111111111111111111111111111111111111111111111111111111

Figure 1: Royal Road function $R_1$.

In previous work we have developed a class of fitness landscapes (the "Royal Road" functions; Mitchell, Forrest, & Holland, 1992; Forrest & Mitchell, 1993) designed to be the simplest class containing the features that are most relevant to the performance of the GA. One of our purposes in developing these landscapes is to carry out systematic comparisons with other search methods.

A simple Royal Road function, $R_1$, is shown in Figure 1. $R_1$ consists of a list of partially specified bit strings (*schemas*) $s_i$ in which '*' denotes a wild card (either 0 or 1). Each schema $s_i$ is given with a coefficient $c_i$. The *order* of a schema is the number of defined (non-'*') bits. A bit string $x$ is said to be an *instance* of a schema $s$, $x \in s$, if $x$ matches $s$ in the defined positions. The fitness $R_1(x)$ of a bit string $x$ is defined as follows:

$$R_1(x) = \sum_i c_i \delta_i(x), \text{where } \delta_i(x) = \begin{cases} 1 & \text{if } x \in s_i \\ 0 & \text{otherwise.} \end{cases}$$

For example, if $x$ is an instance of exactly two of the order-8 schemas, $R_1(x) = 16$. Likewise, $R_1(111 \ldots 1) = 64$.

The Building Block Hypothesis (Holland, 1975/1992) states that the GA works well when instances of low-order, short schemas ("building blocks") that confer high fitness can be recombined to form instances of larger schemas that confer even higher fitness. Given this hypothesis, we initially expected that the building-block structure of $R_1$ would lay out a "royal road" for the GA to follow to the optimal string. We also expected that simple hill-climbing schemes would perform poorly since a large number of bit positions must be optimized simultaneously in order to move from an instance of a lower-order schema (e.g., 11111111**...*) to an instance of a higher-order intermediate schema (e.g., 11111111********11111111**...*). However both these expectations were overturned (Forrest & Mitchell, 1993). In our experiments, a simple GA (using fitness-proportionate selection with sigma scaling, single-point crossover, and point mutation) optimized $R_1$ quite slowly, at least in part because of "hitchhiking": once an instance of a higher-order schema is discovered, its high fitness allows the schema to spread quickly in the population, with 0s in other positions in the string hitchhiking along with the 1s in the schema's defined positions. This slows down the discovery of schemas in the other positions, especially those that are close to the highly fit schema's defined positions. Hitchhiking can in general be a serious bottleneck for the GA, and we observed similar effects

| 200 runs | GA | SAHC | NAHC | RMHC |
|----------|-----|------|------|------|
| Mean | 61,334 (2304) | > 256,000 (0) | > 256,000 (0) | 6179 (186) |
| Median | 54,208 | > 256,000 | > 256,000 | 5775 |

Table 1: Mean and median number of function evaluations to find the optimum string over 200 runs of the GA and of various hill-climbing algorithms on $R_1$. The standard error is given in parentheses.

in several variations of our original GA.

Our other expectation—that the GA would outperform simple hill-climbing on these functions—was also proved wrong. Forrest and Mitchell (1993) compared the GA's performance on a variation of $R_1$ with three different hill-climbing methods: steepest ascent hill-climbing (SAHC), next-ascent hill-climbing (NAHC), and a zero-temperature Monte Carlo method, which Forrest and Mitchell called "random mutation hill-climbing" (RMHC). In RMHC, a string is chosen at random and its fitness is evaluated. The string is then mutated at a randomly chosen single locus, and the new fitness is evaluated. If the mutation leads to an equal or higher fitness, the new string replaces the old string. This procedure is iterated until the optimum has been found or a maximum number of function evaluations has been performed.

Here we have repeated these experiments for $R_1$. The results (similar to those given for $R_2$ in Forrest & Mitchell, 1993) are given in Table 1. We compare the mean and median number of function evaluations to find the optimum string rather than mean and median absolute run time, because in almost all GA applications (e.g., evolving neural-network architectures), the time to perform a function evaluation vastly dominates the time required to execute other parts of the algorithm. For this reason, we consider all parts of the algorithm excluding the function evaluations to take negligible time.

The results on SAHC and NAHC were as expected—while the GA found the optimum on $R_1$ in an average of 61,334 function evaluations, neither SAHC nor NAHC ever found the optimum within the maximum of 256,000 function evaluations. However, RMHC found the optimum on $R_1$ in an average of 6179 function evaluations—nearly a factor of ten faster than the GA. This striking difference on landscapes originally designed to be "royal roads" for the GA underscores the need for a rigorous answer to the question posed earlier: "Under what conditions will a GA outperform other search algorithms, such as hill climbing?"

## 2 ANALYSIS OF RMHC AND AN IDEALIZED GA

To begin to answer this question, we analyzed the RMHC algorithm with respect to $R_1$. Suppose the fitness function consists of $N$ adjacent blocks of $K$ 1s each (in $R_1$, $N = 8$ and $K = 8$). What is the expected time (number of function evaluations) $E(K, N)$ to find the optimum string of all 1s? We can first ask a simpler question: what is the expected time $E(K, 1)$ to find a single block of $K$ 1s? A Markov-chain analysis (not given here) yields $E(K, 1)$ slightly larger than $2^K$, converging slowly to $2^K$ from above as $K \to \infty$ (Richard Palmer, personal communication). For

example, for $K = 8$, $E(K, 1) = 301.2$.

Now suppose we want RMHC to discover a string with $N$ blocks of $K$ 1s. The time to discover a first block of $K$ 1s is $E(K, 1)$, but, once it has been found, the time to discover a second block is longer, since many of the function evaluations are "wasted" on testing mutations inside the first block. The proportion of *non-wasted* mutations is $(KN - K)/KN$; this is the proportion of mutations that occur in the $KN - K$ positions outside the first block. The expected time $E(K, 2)$ to find a second block is $E(K, 1) + E(K, 1)[KN/(KN - K)]$. Similarly, the total expected time is:

$$
\begin{aligned}
E(K, N) &= E(K, 1) + E(K, 1)\frac{N}{N - 1} + \ldots + E(K, 1)\frac{N}{N - (N - 1)} \\
&= E(K, 1)N\left[1 + \frac{1}{2} + \frac{1}{3} + \ldots + \frac{1}{N}\right].
\end{aligned}
\tag{1}
$$

(The actual value may be a bit larger, since E(K,1) is the expected time to the first block, whereas $E(K, N)$ depends on the worst time for the $N$ blocks.) Expression (1) is approximately $E(K, 1)N(log N + \gamma)$, where $\gamma$ is Euler's constant. For $K = 8, N = 8$, the value of expression (1) is 6549. When we ran RMHC on the $R_1$ function 200 times, the average number of function evaluations to the optimum was 6179, which agrees reasonably well with the expected value.

Could a GA ever do better than this? There are three reasons why we might expect a GA to perform well on $R_1$. First, at least theoretically the GA is fast because of *implicit parallelism* (Holland, 1975/1992): each string in the population is an instance of many different schemas, and if the population is large enough and is initially chosen at random, a large number of different schemas—many more than the number of strings in the population—are being sampled in parallel. This should result in a quick search for short, low-order schemas that confer high fitness. Second, fitness-proportionate reproduction under the GA should conserve instances of such schemas. Third, a high crossover rate should quickly combine instances of low-order schemas on different strings to create instances of longer schemas that confer even higher fitness. Our previous experiments (Forrest & Mitchell, 1993) showed that the simple GA departed from this "in principle" behavior. One major impediment was hitchhiking, which limited implicit parallelism by fixing certain schema regions suboptimally. But if the GA worked exactly as described above, how quickly could it find the optimal string of $R_1$?

To answer this question we consider an "idealized genetic algorithm" (IGA) that explicitly has the features described above. The IGA knows ahead of time what the desired schemas are, and a "function evaluation" is the determination of whether a given string contains one or more of them. In the IGA, at each time step a single string is chosen at random, with uniform probability for each bit. The string is "evaluated" by determining whether it is an instance of one or more of the desired schemas. The first time such a string is found, it is sequestered. At each subsequent discovery of an instance of one or more not-yet-discovered schemas the new string is instantaneously crossed over with the sequestered string so that the sequestered string contains all the desired schemas that have been discovered so far.

This procedure is unusable in practice, since it requires knowing *a priori* which schemas are relevant, whereas in general an algorithm such as the GA or RMHC

directly measures the fitness of a string, and does not know ahead of time which schemas contribute to high fitness. However, the idea behind the GA is to do implicitly what the IGA is able to do explicitly. This idea will be elaborated below.

Suppose again that our desired schemas consist of $N$ blocks of $K$ 1s each. What is the expected time (number of function evaluations) until the saved string contains all the desired schemas? Solutions have been suggested by G. Huber (personal communication), and A. Shevoroskin (personal communication), and a detailed solution is given in (Holland, 1993). The main idea is to note that the probability of finding a single desired block $s$ on a random string is $p = 1/2^K$, and the probability of finding $s$ by time $t$ is $1 - (1-p)^t$. Then the probability $\mathcal{P}_N(t)$ that all $N$ blocks have been found by time $t$ is:

$$\mathcal{P}_N(t) = (1 - (1-p)^t)^N,$$

and the probability $P_N(t)$ that all $N$ blocks are found at exactly time $t$ is:

$$P_N(t) = [1 - (1-p)^t]^N - [1 - (1-p)^{t-1}]^N.$$

The expected time is then

$$E_N = \sum_1^\infty t \left([1 - (1-p)^t]^N - [1 - (1-p)^{t-1}]^N\right).$$

This sum can be expanded and simplified, and with some work, along with the approximation $(1-p)^n \approx 1 - np$ for small $p$, we obtain the following approximation:

$$E_N \approx (1/p) \sum_{n=1}^N \frac{1}{n} \approx 2^K(\log N + \gamma).$$

The major point is that the IGA gives an expected time that is on the order of $2^K \log N$, where RMHC gives an expected time that is on the order of $2^K N \log N$, a factor of $N$ slower. This kind of analysis can help us predict how and when the GA will outperform hill climbing.

What makes the IGA faster than RMHC? A primary reason is that the IGA perfectly implements implicit parallelism: each new string is completely independent of the previous one, so new samples are given independently to each schema region. In contrast, RMHC moves in the space of strings by single-bit mutations from an original string, so each new sample has all but one of the same bits as the previous sample. Thus each new string gives a new sample to only one schema region. The IGA spends more time than RMHC constructing new samples, but since we are counting only function evaluations, we ignore the construction time. The IGA "cheats" on each function evaluation, since it knows exactly the desired schemas, but in this way it gives a lower bound on the number of function evaluations that the GA will need on this problem.

Independent sampling allows for a speed-up in the IGA in two ways: it allows for the possibility of more than one desirable schema appearing simultaneously on a given sample, and it also means that there are no wasted samples as there are in RMHC. Although the comparison we have made is with RMHC, the IGA will also be significantly faster on $R_1$ (and similar landscapes) than any hill-climbing

Level 1: $s_1$ $s_2$ $\quad$ $s_3$ $s_4$ $\quad$ $s_5$ $s_6$ $\quad$ $s_7$ $s_8$ $\quad$ $s_9$ $s_{10}$ $\quad$ $s_{11}$ $s_{12}$ $\quad$ $s_{13}$ $s_{14}$ $\quad$ $s_{15}$ $s_{16}$
Level 2: $(s_1\ s_2)\ (s_3\ s_4)\ (s_5\ s_6)\ (s_7\ s_8)\ (s_9\ s_{10})\ (s_{11}\ s_{12})\ (s_{13}\ s_{14})\ (s_{15}\ s_{16})$
Level 3: $(s_1\ s_2 \quad s_3\ s_4)\ (s_5\ s_6 \quad s_7\ s_8)\ (s_9\ s_{10} \quad s_{11}\ s_{12})\ (s_{13}\ s_{14} \quad s_{15}\ s_{16})$
Level 4: $(s_1\ s_2 \quad s_3\ s_4 \quad s_5\ s_6 \quad s_7\ s_8)\ (s_9\ s_{10} \quad s_{11}\ s_{12} \quad s_{13}\ s_{14} \quad s_{15}\ s_{16})$

Figure 2: Royal Road Function $R4$.

method that works by mutating single bits (or a small number of bits) to obtain new samples.

The hitchhiking effects described earlier also result in a loss of independent samples for the real GA. The goal is to have the real GA, as much as possible, approximate the IGA. Of course, the IGA works because it explicitly knows what the desired schemas are; the real GA does not have this information and can only estimate what the desired schemas are by an implicit sampling procedure. But it is possible for the real GA to approximate a number of the features of the IGA. *Independent samples:* The population size has to be large enough, the selection process has to be slow enough, and the mutation rate has to be sufficient to make sure that no single locus is fixed at a single value in every (or even a large majority) of strings in the population. *Sequestering desired schemas:* Selection has to be strong enough to preserve desired schemas that have been discovered, but it also has to be slow enough (or, equivalently, the relative fitness of the non-overlapping desirable schemas has to be small enough) to prevent significant hitchhiking on some highly fit schemas, which can crowd out desired schemas in other parts of the string. *Instantaneous crossover:* The crossover rate has to be such that the time for a crossover to occur that combines two desired schemas is small with respect to the discovery time for the desired schemas. *Speed-up over RMHC:* The string length (a function of $N$) has to be large enough to make the $N$ speed-up factor significant.

These mechanisms are not all mutually compatible (e.g., high mutation works against sequestering schemas), and thus must be carefully balanced against one another. A discussion of how such a balance might be achieved is given in Holland (1993).

## 3   RESULTS OF EXPERIMENTS

As a first step in exploring these balances, we designed $R3$, a variant of our previous function $R_2$ (Forrest & Mitchell, 1993), based on some of the features described above. In $R3$ the desired schemas are $s_1$–$s_8$ (shown in Fig. 1) and combinations of them, just as in $R2$. However, in $R3$ the lowest-level order-8 schemas are each separated by "introns" (bit positions that do not contribute to fitness—see Forrest & Mitchell, 1993; Levenick, 1991) of length 24.

In $R3$, a string that is not an instance of any desired schema receives fitness 1.0. Every time a new level is reached—i.e., a string is found that is an instance of one or more schemas at that level—a small increment $u$ is added to the fitness. Thus strings at level 1 (that are instances of at least one level-1 schema) have fitness $1 + u$, strings at level 2 have fitness $1 + 2u$, etc. For our experiments we set $u = 0.2$.

|  |  | Level 1 | Level 2 | Level 3 |
|---|---|---|---|---|
| GA | evals | 500 (0) | 4486 (478) | 86,078 (17,242) |
|  | % runs | 100 | 100 | 86 |
| RMHC | evals | 230 (36) | 8619 (1013) | 95,027 (17,948) |
|  | % runs | 100 | 100 | 41 |

Table 2: *R*4: Mean function evaluations (over 37 runs) to attain each level for the GA and for RMHC. In the GA runs, the number of function evaluations is sampled every 500 evaluations, so each value is actually an upper bound for an interval of length 500. The standard errors are in parentheses. The percentage of runs which reached each level is shown next to the heading "% runs." Only runs which successfully reached a given level were included in the function evaluation calculations for that level.

The purpose of the introns was to help maintain independent samples in each schema position by preventing linkage between schema positions. The independence of samples was also helped by using a larger population (2000) and the much slower selection scheme given by the function. In preliminary experiments on *R*3 (not shown) hitchhiking in the GA was reduced significantly, and the population was able to maintain instances of all the lowest-level schemas throughout each run.

Next, we studied *R*4 (illustrated in Figure 2). *R*4 is identical to *R*3, except that it does not have introns. Further, *R*4 is defined over 128-bit strings, thus doubling the size of the problem. In preliminary runs on *R*4, we used a population size of 500, a mutation rate of 0.005 (mutation always flips a bit), and multipoint crossover, where the number of crossover points for each pair of parents was selected from a Poisson distribution with mean 2.816.

Table 2 gives the mean number of evaluations to reach levels 1, 2, and 3 (neither algorithm reached level 4 within the maximum of $10^6$ function evaluations). As can be seen, the time to reach level one is comparable for the two algorithms, but the GA is much faster at reaching levels 2 and 3. Further, the GA discovers level 3 approximately twice as often as RMHC. As was said above, it is necessary to balance the maintenance of independent samples with the sequestering of desired schemas. These preliminary results suggest that *R*4 does a better job of maintaining this balance than the earlier Royal Road functions. Working out these balances in greater detail is a topic of future work.

## 4 CONCLUSION

We have presented analyses of two algorithms, RMHC and the IGA, and have used the analyses to identify some general principles of when and how a genetic algorithm will outperform hill climbing. We then presented some preliminary experimental results comparing the GA and RMHC on a modified Royal Road landscape. These analyses and results are a further step in achieving our original goals—to design the simplest class of fitness landscapes that will distinguish the GA from other search methods, and to characterize rigorously the general features of a fitness landscape that make it suitable for a GA.

Our modified Royal Road landscape $R4$, like $R_1$, is not meant to be a realistic example of a problem to which one might apply a GA. Rather, it is meant to be an idealized problem in which certain features most relevant to GAs are explicit, so that the GA's performance can be studied in detail. Our claim is that in order to understand how the GA works in general and where it will be most useful, we must first understand how it works and where it will be most useful on simple yet carefully designed landscapes such as these. The work reported here is a further step in this direction.

## Acknowledgments

We thank R. Palmer for suggesting the RMHC algorithm and for sharing his careful analysis with us, and G. Huber for his assistance on the analysis of the IGA. We also thank E. Baum, L. Booker, T. Jones, and R. Riolo for helpful comments and discussions regarding this work. We gratefully acknowledge the support of the Santa Fe Institute's Adaptive Computation Program, the Alfred P. Sloan Foundation (grant B1992-46), and the National Science Foundation (grants IRI-9157644 and IRI-9224912).

## References

L. D. Davis (1991). Bit-climbing, representational bias, and test suite design. In R. K. Belew and L. B. Booker (eds.), *Proceedings of the Fourth International Conference on Genetic Algorithms*, 18–23. San Mateo, CA: Morgan Kaufmann.

K. A. De Jong (1975). *An Analysis of the Behavior of a Class of Genetic Adaptive Systems.* Unpublished doctoral dissertation. University of Michigan, Ann Arbor, MI.

S. Forrest and M. Mitchell (1993). Relative building-block fitness and the building-block hypothesis. In D. Whitley (ed.), *Foundations of Genetic Algorithms 2*, 109–126. San Mateo, CA: Morgan Kaufmann.

J. H. Holland (1975/1992). *Adaptation in Natural and Artificial Systems.* Cambridge, MA: MIT Press. (First edition 1975, Ann Arbor: University of Michigan Press.)

J. H. Holland (1993). *Innovation in complex adaptive systems: Some mathematical sketches.* Working Paper 93-10-062, Santa Fe Institute, Santa Fe, NM.

L. Ingber and B. Rosen (1992). Genetic algorithms and very fast simulated reannealing: A comparison. *Mathematical Computer Modelling, 16 (11)*, 87–100.

J. R. Levenick (1991). Inserting introns improves genetic algorithm success rate: Taking a cue from biology. In R. K. Belew and L. B. Booker (eds.), *Proceedings of the Fourth International Conference on Genetic Algorithms*, 123–127. San Mateo, CA: Morgan Kaufmann.

M. Mitchell, S. Forrest, and J. H. Holland (1992). The royal road for genetic algorithms: Fitness landscapes and GA performance. In F. J. Varela and P. Bourgine (eds.), *Proceedings of the First European Conference on Artificial Life*, 245–254. Cambridge, MA: MIT Press.
